# Nonlinear Inverse Reinforcement Learning with Gaussian Processes

**Sergey Levine**
Stanford University
svlevine@cs.stanford.edu

**Zoran Popović**
University of Washington
zoran@cs.washington.edu

**Vladlen Koltun**
Stanford University
vladlen@cs.stanford.edu

## Abstract

We present a probabilistic algorithm for nonlinear inverse reinforcement learning. The goal of inverse reinforcement learning is to learn the reward function in a Markov decision process from expert demonstrations. While most prior inverse reinforcement learning algorithms represent the reward as a linear combination of a set of features, we use Gaussian processes to learn the reward as a nonlinear function, while also determining the relevance of each feature to the expert's policy. Our probabilistic algorithm allows complex behaviors to be captured from suboptimal stochastic demonstrations, while automatically balancing the simplicity of the learned reward structure against its consistency with the observed actions.

## 1 Introduction

Inverse reinforcement learning (IRL) methods learn a reward function in a Markov decision process (MDP) from expert demonstrations, allowing the expert's policy to be generalized to unobserved situations [7]. Each task is consistent with many reward functions, but not all rewards provide a compact, portable representation of the task, so the central challenge in IRL is to find a reward with meaningful structure [7]. Many prior methods impose structure by describing the reward as a linear combination of hand selected features [1, 12]. In this paper, we extend the Gaussian process model to learn highly nonlinear reward functions that still compactly capture the demonstrated behavior.

GP regression requires input-output pairs [11], and was previously used for value function approximation [10, 4, 2]. Our Gaussian Process Inverse Reinforcement Learning (GPIRL) algorithm only observes the expert's actions, not the rewards, so we extend the GP model to account for the stochastic relationship between actions and underlying rewards. This allows GPIRL to balance the simplicity of the learned reward function against its consistency with the expert's actions, without assuming the expert to be optimal. The learned GP kernel hyperparameters capture the structure of the reward, including the relevance of each feature. Once learned, the GP can recover the reward for the current state space, and can predict the reward for any unseen state space within the domain of the features.

Previous IRL algorithms generally learn the reward as a linear combination of features, either by finding a reward under which the expert's policy has a higher value than all other policies by a margin [7, 1, 12, 15], or else by maximizing the probability of the reward under a model of near-optimal expert behavior [6, 9, 17, 3]. While several margin-based methods learn nonlinear reward functions through feature construction [13, 14, 5], such methods assume optimal expert behavior. To the best of our knowledge, GPIRL is the first method to combine probabilistic reasoning about stochastic expert behavior with the ability to learn the reward as a nonlinear function of features, allowing it to outperform prior methods on tasks with inherently nonlinear rewards and suboptimal examples.

## 2   Inverse Reinforcement Learning Preliminaries

A Markov decision process is defined as a tuple $\mathcal{M} = \{\mathcal{S}, \mathcal{A}, \mathcal{T}, \gamma, \mathbf{r}\}$, where $\mathcal{S}$ is the state space, $\mathcal{A}$ is the set of actions, $\mathcal{T}_{s'}^{sa}$ is the probability of a transition from $s \in \mathcal{S}$ to $s' \in \mathcal{S}$ under action $a \in \mathcal{A}$, $\gamma \in [0, 1)$ is the discount factor, and $\mathbf{r}$ is the reward function. The optimal policy $\pi^\star$ maximizes the expected discounted sum of rewards $E\left[\sum_{t=0}^{\infty} \gamma^t \mathbf{r}_{s_t} | \pi^\star\right]$. In inverse reinforcement learning, the algorithm is presented with $\mathcal{M} \setminus \mathbf{r}$, as well as expert demonstrations, denoted $\mathcal{D} = \{\zeta_1, ..., \zeta_N\}$, where $\zeta_i$ is a path $\zeta_i = \{(s_{i,0}, a_{i,0}), ..., (s_{i,T}, a_{i,T})\}$. The algorithm is also presented with features of the form $f : \mathcal{S} \to \mathbb{R}$ that can be used to represent the unknown reward $\mathbf{r}$.

IRL aims to find a reward function $\mathbf{r}$ under which the optimal policy matches the expert's demonstrations. To this end, we could assume that the examples $\mathcal{D}$ are drawn from the optimal policy $\pi^\star$. However, real human demonstrations are rarely optimal. One approach to learning from a suboptimal expert is to use a probabilistic model of the expert's behavior. We employ the maximum entropy IRL (MaxEnt) model [17], which is closely related to linearly-solvable MDPs [3], and has been used extensively to learn from human demonstrations [16, 17]. Under this model, the probability of taking a path $\zeta$ is proportional to the exponential of the rewards encountered along that path. This model is convenient for IRL, because its likelihood is differentiable [17], and a complete stochastic policy uniquely determines the reward function [3]. Intuitively, such a stochastic policy is more deterministic when the stakes are high, and more random when all choices have similar value.

Under this policy, the probability of an action $a$ in state $s$ can be shown to be proportional to the exponential of the expected total reward after taking the action, denoted $P(a|s) \propto \exp(\mathbf{Q}_{sa}^{\mathbf{r}})$, where $\mathbf{Q}^{\mathbf{r}} = \mathbf{r} + \gamma \mathcal{T} \mathbf{V}^{\mathbf{r}}$. The value function $\mathbf{V}^{\mathbf{r}}$ is computed with a "soft" version of the familiar Bellman backup operator: $\mathbf{V}_s^{\mathbf{r}} = \log \sum_a \exp \mathbf{Q}_{sa}^{\mathbf{r}}$. The probability of $a$ in state $s$ is therefore normalized by $\exp \mathbf{V}^{\mathbf{r}}$, giving $P(a|s) = \exp(\mathbf{Q}_{sa}^{\mathbf{r}} - \mathbf{V}_s^{\mathbf{r}})$. Detailed derivations of these equations can be found in prior work [16]. The complete log likelihood of the data under $\mathbf{r}$ can be written as

$$\log P(\mathcal{D}|\mathbf{r}) = \sum_i \sum_t \log P(a_{i,t}|s_{i,t}) = \sum_i \sum_t \left( \mathbf{Q}_{s_{i,t}a_{i,t}}^{\mathbf{r}} - \mathbf{V}_{s_{i,t}}^{\mathbf{r}} \right) \tag{1}$$

While we can maximize Equation 1 directly to obtain $\mathbf{r}$, such a reward is unlikely to exhibit meaningful structure, and would not be portable to novel state spaces. Prior methods address this problem by representing $\mathbf{r}$ as a linear combination of a set of provided features [17]. However, if $\mathbf{r}$ is not linear in the features, such methods are not sufficiently expressive. In the next section, we describe how Gaussian processes can be used to learn $\mathbf{r}$ as a general nonlinear function of the features.

## 3   The Gaussian Process Inverse Reinforcement Learning Algorithm

GPIRL represents the reward as a nonlinear function of feature values. This function is modeled as a Gaussian process, and its structure is determined by its kernel function. The Bayesian GP framework provides a principled method for learning the hyperparameters of this kernel, thereby learning the structure of the unknown reward. Since the reward is not known, we use Equation 1 to specify a distribution over GP outputs, and learn both the output values and the kernel function.

In GP regression, we use noisy observations $\mathbf{y}$ of the true underlying outputs $\mathbf{u}$. GPIRL directly learns the true outputs $\mathbf{u}$, which represent the rewards associated with feature coordinates $\mathbf{X_u}$. These coordinates may simply be the feature values of all states or, as discussed in Section 5, a subset of all states. The rewards at states that are not included in this subset are inferred by the GP. We also learn the kernel hyperparameters $\boldsymbol{\theta}$ in order to recover the structure of the reward. The most likely values of $\mathbf{u}$ and $\boldsymbol{\theta}$ are found by maximizing their probability under the expert demonstrations $\mathcal{D}$:

$$P(\mathbf{u}, \boldsymbol{\theta}|\mathcal{D}, \mathbf{X_u}) \propto P(\mathcal{D}, \mathbf{u}, \boldsymbol{\theta}|\mathbf{X_u}) = \left[ \int_{\mathbf{r}} \underbrace{P(\mathcal{D}|\mathbf{r})}_{\text{IRL term}} \underbrace{P(\mathbf{r}|\mathbf{u}, \boldsymbol{\theta}, \mathbf{X_u})}_{\text{GP posterior}} d\mathbf{r} \right] \underbrace{P(\mathbf{u}, \boldsymbol{\theta}|\mathbf{X_u})}_{\text{GP probability}} \tag{2}$$

The log of $P(\mathcal{D}|\mathbf{r})$ is given by Equation 1, the GP posterior $P(\mathbf{r}|\mathbf{u}, \boldsymbol{\theta}, \mathbf{X_u})$ is the probability of a reward function under the current values of $\mathbf{u}$ and $\boldsymbol{\theta}$, and $P(\mathbf{u}, \boldsymbol{\theta}|\mathbf{X_u})$ is the prior probability of a

particular assignment to $\mathbf{u}$ and $\boldsymbol{\theta}$. The log of $P(\mathbf{u}, \boldsymbol{\theta}|\mathbf{X_u})$ is the GP log marginal likelihood, which favors simple kernel functions and values of $\mathbf{u}$ that conform to the current kernel matrix [11]:

$$\log P(\mathbf{u}, \boldsymbol{\theta}|\mathbf{X_u}) = -\frac{1}{2}\mathbf{u}^{\mathrm{T}}\mathbf{K}_{\mathbf{u},\mathbf{u}}^{-1}\mathbf{u} - \frac{1}{2}\log|\mathbf{K}_{\mathbf{u},\mathbf{u}}| - \frac{n}{2}\log 2\pi + \log P(\boldsymbol{\theta}) \quad (3)$$

The last term $\log P(\boldsymbol{\theta})$ is a hyperparameter prior, which is discussed in Section 4. The entries of the covariance matrix $\mathbf{K}_{\mathbf{u},\mathbf{u}}$ are given by the kernel function. In order to determine the relevance of each feature, we use the automatic relevance detection (ARD) kernel, with hyperparameters $\boldsymbol{\theta} = \{\beta, \mathbf{\Lambda}\}$:

$$k(\mathbf{x}_i, \mathbf{x}_j) = \beta \exp\left(-\frac{1}{2}(\mathbf{x}_i - \mathbf{x}_j)^{\mathrm{T}}\mathbf{\Lambda}(\mathbf{x}_i - \mathbf{x}_j)\right)$$

The hyperparameter $\beta$ is the overall variance, and the diagonal matrix $\mathbf{\Lambda}$ specifies the weight on each feature. When $\mathbf{\Lambda}$ is learned, less relevant features receive low weights, and more relevant features receive high weights. States distinguished by highly-weighted features can take on different reward values, while those that have similar values for all highly-weighted features take on similar rewards.

The GP posterior $P(\mathbf{r}|\mathbf{u}, \boldsymbol{\theta}, \mathbf{X_u})$ is a Gaussian distribution with mean $\mathbf{K}_{\mathbf{r},\mathbf{u}}^{\mathrm{T}}\mathbf{K}_{\mathbf{u},\mathbf{u}}^{-1}\mathbf{u}$ and covariance $\mathbf{K}_{\mathbf{r},\mathbf{r}} - \mathbf{K}_{\mathbf{r},\mathbf{u}}^{\mathrm{T}}\mathbf{K}_{\mathbf{u},\mathbf{u}}^{-1}\mathbf{K}_{\mathbf{r},\mathbf{u}}$. $\mathbf{K}_{\mathbf{r},\mathbf{u}}$ is the covariance of the rewards at all states with the inducing point values $\mathbf{u}$, located respectively at $\mathbf{X_r}$ and $\mathbf{X_u}$ [11]. Due to the complexity of $P(\mathcal{D}|\mathbf{r})$, the integral in Equation 2 cannot be computed in closed form. Instead, we can consider this problem as analogous to sparse approximation for GP regression [8], where a small set of inducing points $\mathbf{u}$ acts as the support for the full set of training points $\mathbf{r}$. In this context, the Gaussian posterior distribution over $\mathbf{r}$ is called the training conditional. One approximation is to assume that the training conditional is deterministic – that is, has variance zero [8]. This approximation is particularly appropriate in our case, because if the learned GP is used to predict a reward for a novel state space, the most likely reward would have the same form as the mean of the training conditional. Under this approximation, the integral disappears, and $\mathbf{r}$ is set to $\mathbf{K}_{\mathbf{r},\mathbf{u}}^{\mathrm{T}}\mathbf{K}_{\mathbf{u},\mathbf{u}}^{-1}\mathbf{u}$. The resulting log likelihood is simply

$$\log P(\mathcal{D}, \mathbf{u}, \boldsymbol{\theta}|\mathbf{X_u}) = \underbrace{\log P(\mathcal{D}|\mathbf{r} = \mathbf{K}_{\mathbf{r},\mathbf{u}}^{\mathrm{T}}\mathbf{K}_{\mathbf{u},\mathbf{u}}^{-1}\mathbf{u})}_{\text{IRL log likelihood}} + \underbrace{\log P(\mathbf{u}, \boldsymbol{\theta}|\mathbf{X_u})}_{\text{GP log likelihood}} \quad (4)$$

Once the likelihood is optimized, the reward $\mathbf{r} = \mathbf{K}_{\mathbf{r},\mathbf{u}}^{\mathrm{T}}\mathbf{K}_{\mathbf{u},\mathbf{u}}^{-1}\mathbf{u}$ can be used to recover the expert's policy on the entire state space. The GP can also predict the reward function for any novel state space in the domain of the features. The most likely reward for a novel state space is the mean posterior $\mathbf{K}_{\star,\mathbf{u}}^{\mathrm{T}}\mathbf{K}^{-1}\mathbf{u}$, where $\mathbf{K}_{\star,\mathbf{u}}$ is the covariance of the new states and the inducing points. In our implementation, the likelihood is optimized with the L-BFGS method, with derivatives provided in the supplement. When the hyperparameters are learned, the likelihood is generally not convex. While this is not unusual for GP methods, it does mean that the method can suffer from local optima. In the supplement, we also describe a simple restart procedure we used to mitigate this problem.

## 4 Regularization and Hyperparameter Priors

In GP regression, a noise term is often added to the diagonal of the kernel matrix to account for noisy observations. Since GPIRL learns the noiseless underlying outputs $\mathbf{u}$, there is no cause to add a noise term, which means that the kernel matrix $\mathbf{K}_{\mathbf{u},\mathbf{u}}$ can become singular. Intuitively, this indicates that two or more inducing points are deterministically covarying, and therefore redundant. To ensure that no inducing point is redundant, we assume that their positions in feature space $\mathbf{X_u}$, rather than their values, are corrupted by white noise with variance $\sigma^2$. The expected squared difference in the $k^{\text{th}}$ feature values between two points $\mathbf{x}_i$ and $\mathbf{x}_j$ is then given by $(x_{ik} - x_{jk})^2 + 2\sigma^2$, and the new, regularized kernel function is given by

$$k(\mathbf{x}_i, \mathbf{x}_j) = \beta \exp\left(-\frac{1}{2}(\mathbf{x}_i - \mathbf{x}_j)^{\mathrm{T}}\mathbf{\Lambda}(\mathbf{x}_i - \mathbf{x}_j) - 1_{i \neq j}\sigma^2\text{tr}(\mathbf{\Lambda})\right) \quad (5)$$

The regularization ensures that $k(\mathbf{x}_i, \mathbf{x}_j) < k(\mathbf{x}_i, \mathbf{x}_i)$ so long as at least one feature is relevant – that is, $\mathrm{tr}(\mathbf{\Lambda}) > 0$. While the regularized kernel prevents singular covariance matrices when many features become irrelevant, the log likelihood can still increase to infinity as $\mathbf{\Lambda} \to 0$ or $\beta \to 0$: in both cases, $-\frac{1}{2}\log|\mathbf{K}_{\mathbf{u},\mathbf{u}}| \to \infty$ and, so long as $\mathbf{u} \to 0$, all other terms remain finite.

To prevent such degeneracies, we use a hyperparameter prior that discourages kernels under which two inducing points become deterministically covarying. As two points $\mathbf{u}_i$ and $\mathbf{u}_j$ become deterministically related, the magnitude of their partial correlation $[\mathbf{K}_{\mathbf{u},\mathbf{u}}^{-1}]_{ij}$ becomes infinity. We can therefore prevent degeneracies with a prior term of the form $-\frac{1}{2}\sum_{ij}[\mathbf{K}_{\mathbf{u},\mathbf{u}}^{-1}]_{ij}^2 = -\frac{1}{2}\mathrm{tr}(\mathbf{K}_{\mathbf{u},\mathbf{u}}^{-2})$, which discourages large partial correlations between inducing points. Such a prior is dependent on $\mathbf{X_u}$. However, unlike in GP regression, $\mathbf{X_u}$ and $\mathbf{u}$ are parameters of the algorithm rather than data, and since the inducing point positions are fixed in advance, it is possible to condition the prior on $\mathbf{X_u}$.

To encourage sparse feature weights $\mathbf{\Lambda}$, we also use a sparsity-inducing penalty $\phi(\mathbf{\Lambda})$, resulting in the prior $\log P(\boldsymbol{\theta}|\mathbf{X_u}) = -\frac{1}{2}\mathrm{tr}(\mathbf{K}_{\mathbf{u},\mathbf{u}}^{-2}) - \phi(\mathbf{\Lambda})$. A variety of penalties are suitable, but we obtained the best results with $\phi(\mathbf{\Lambda}) = \sum_i \log(\mathbf{\Lambda}_{ii}+1)$. Although we can also optimize for the noise variance $\sigma^2$, we did not observe that this significantly altered the results, and instead fixed $2\sigma^2$ to $10^{-2}$.

## 5 Inducing Points and Large State Spaces

A straightforward choice for the inducing points $\mathbf{X_u}$ is the feature values of all states in the state space $\mathcal{S}$. Unfortunately, the kernel matrix $\mathbf{K}_{\mathbf{u},\mathbf{u}}$ is constructed and inverted at each iteration of the optimization in order to compute the gradient. This is a costly procedure: constructing the matrix has running time $\mathcal{O}(d_{\mathbf{X}}|\mathbf{X_u}|^2)$ and inverting it is $\mathcal{O}(|\mathbf{X_u}|^3)$, where $d_{\mathbf{X}}$ is the number of features. To make GPIRL tractable on large state spaces, we can instead choose $\mathbf{X_u}$ to be a small subset of $\mathcal{S}$, so that only the construction of $\mathbf{K}_{\mathbf{r},\mathbf{u}}$ depends on $|\mathcal{S}|$, and this dependence is linear. In principle, the minimum size of $\mathbf{X_u}$ corresponds to the complexity of the reward function. For example, if the true reward has two constant regions, it can be represented by just two properly placed inducing points. In practice, $\mathbf{X_u}$ must cover the space of feature values well enough to represent an unknown reward function, but we can nonetheless use many fewer points than there are states in $\mathcal{S}$.

In our implementation, we chose $\mathbf{X_u}$ to contain the feature values of all states visited in the example paths, as well as additional random states added to raise $|\mathbf{X_u}|$ to a desired size. While this heuristic worked well in our experiments, we can also view the choice of $\mathbf{X_u}$ as analogous to the choice of the active set in sparse GP approximation. A number of methods have been proposed for selecting these sets [8], and applying such methods to GPIRL is a promising avenue for future work.

## 6 Alternative Kernels

The particular choice of kernel function influences the structure of the learned reward. The stationary kernel in Equation 5 favors rewards that are smooth with respect to feature values. Other kernels can be used to learn other types of structure. For example, a reward function might have wide regions with uniform values, punctuated by regions of high-frequency variation, as is the case for piecewise constant rewards. A stationary kernel would have difficulty representing such structure. Instead, we can warp each coordinate $x_{ik}$ of $\mathbf{x}_i$ by a function $w_k(x_{ik})$ to give high resolution to one region, and low resolution everywhere else. One such function is a sigmoid centered at $m_k$ and scaled by $\ell_k$:

$$w_k(x_{ik}) = \frac{1}{1 + \exp\left(-\frac{x_{ik}-m_k}{\ell_k}\right)}$$

Replacing $\mathbf{x}_i$ by $w(\mathbf{x}_i)$ in Equation 5, we get a regularized warped kernel of the form

$$k(\mathbf{x}_i, \mathbf{x}_j) = \beta \exp\left(-\frac{1}{2}\sum_k \mathbf{\Lambda}_{kk}\left[(w_k(x_{ik}) - w_k(x_{jk}))^2 + 1_{i\neq j}\sigma^2(w_k^\sigma(x_{ik}) + w_k^\sigma(x_{jk}))\right]\right)$$

The second term in the sum is the contribution of the noise to the expected distance. Assuming $\sigma^2$ is small, this value can be approximated to first order by setting $w_k^\sigma(x_{ik}) = \frac{\partial w_k}{\partial x_{ik}} + s_k$, where $s_k$ is an

additional parameter that increases the noise in the tails of the sigmoid to prevent degeneracies. The parameters $\mathbf{m}$, $\boldsymbol{\ell}$, and $\mathbf{s}$ are added to $\boldsymbol{\theta}$ and jointly optimized with $\mathbf{u}$ and the other hyperparameters, using unit variance Gaussian priors for $\boldsymbol{\ell}$ and $\mathbf{s}$ and gamma priors for $\mathbf{m}$. Note that this procedure is not equivalent to merely fitting a sigmoid to the reward function, since the reward can still vary nonlinearly in the high resolution regions around each sigmoid center $m_k$. The accompanying supplement includes details about the priors placed on the warp parameters in our implementation, a complete derivation of $w_k^\sigma$, and the derivatives of the warped kernel function.

During the optimization, as the sigmoid scales $\boldsymbol{\ell}$ become small, the derivatives with respect to the sigmoid centers $\mathbf{m}$ fall to zero. If the centers have not yet converged to the correct values, the optimization will end in a local optimum. It is therefore more important to address local optima when using the warped kernel. As mentioned in Section 3, we mitigate the effects of local optima with a small number of random restarts. Details of the particular random restart technique we used can also be found in the supplement.

We presented just one example of how an alternative kernel allows us to learn a reward with a particular structure. Many kernels have been proposed for GPs [11], and this variety of kernel functions can be used to apply GPIRL to new domains and to extend its generality and flexibility.

# 7 Experiments

We compared GPIRL with prior methods on several IRL tasks, using examples sampled from the stochastic MaxEnt policy (see Section 2) as well as human demonstrations. Examples drawn from the stochastic policy can intuitively be viewed as noisy samples of an underlying optimal policy, while the human demonstrations contain the stochasticity inherent in human behavior. GPIRL was compared with the MaxEnt IRL algorithm [17] and FIRL [5], as well as a variant of MaxEnt with a sparsity-inducing Laplace prior, which we refer to as MaxEnt/Lp. We evaluated a variety of other margin-based methods, including Abbeel and Ng's projection algorithm, MMP, MWAL, MMPBoost and LEARCH [1, 12, 15, 13, 14]. Since GPIRL, FIRL, and MaxEnt consistently produced better results, the other algorithms are not shown here, but are included in the supplementary result tables.

We compare the algorithms using the "expected value difference" score, which is a measure of how suboptimal the learned policy is under the true reward. To compute this score, we find the optimal deterministic policy under each learned reward, measure its expected sum of discounted rewards under the true reward function, and subtract this quantity from the expected sum of discounted rewards under the true policy. While we could also evaluate the optimal stochastic policies, this would unfairly penalize margin-based methods, which are unaware of the MaxEnt model. To determine how well each algorithm captured the structure of the reward function, we evaluated the learned reward on the environment on which it was learned, and on 4 additional random environments (denoted "transfer"). Algorithms that do not express the reward function in terms of the correct features are expected to perform poorly on the transfer environments, even if they perform well on the training environment. Methods that correctly identify relevant features should perform well on both. For each environment, we evaluated the algorithms with both discrete and continuous-valued features. In the latter case, GPIRL used the warped kernel in Section 6 and FIRL, which requires discrete features, was not tested. Each test was repeated 8 times with different random environments.

## 7.1 Objectworld Experiments

The objectworld is an $N \times N$ grid of states with five actions per state, corresponding to steps in each direction and staying in place. Each action has a $30\%$ chance of moving in a different random direction. Randomly placed objects populate the objectworld, and each is assigned one of $C$ inner and outer colors. Object placement is randomized in the transfer environments, while $N$ and $C$ remain the same. There are $2C$ continuous features, each giving the Euclidean distance to the nearest object with a specific inner or outer color. In the discrete feature case, there are $2CN$ binary features, each one an indicator for a corresponding continuous feature being less than $d \in \{1, ..., N\}$. The true reward is positive in states that are both within 3 cells of outer color 1 and 2 cells of outer color 2, negative within 3 cells of outer color 1, and zero otherwise. Inner colors and all other outer colors are distractors. The algorithms were provided example paths of length 8, and the number of examples and colors was varied to determine their ability to handle limited data and distractors.

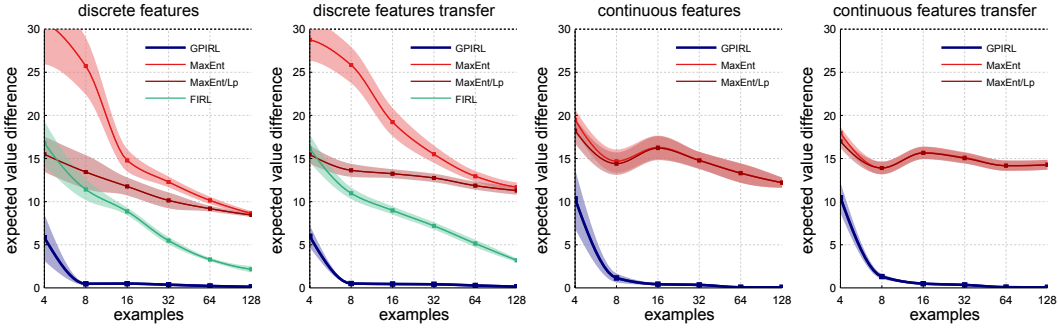

Figure 1: Results for $32 \times 32$ objectworlds with $C = 2$ and varying numbers of examples. Shading shows standard error. GPIRL learned accurate rewards that generalized well to new state spaces.

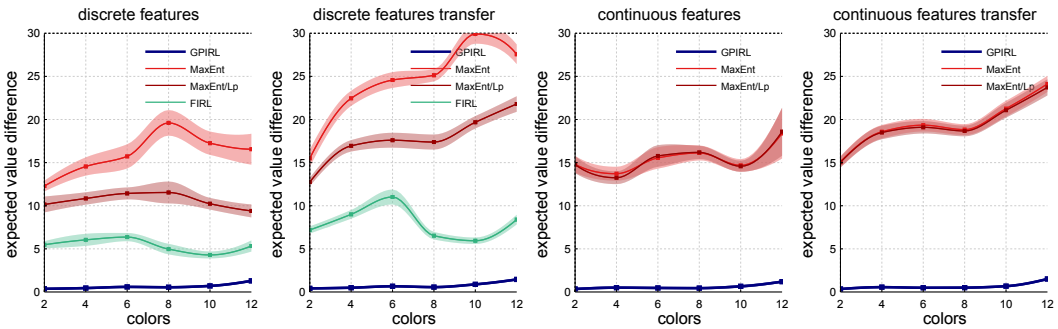

Figure 2: Objectworld evaluation with 32 examples and varying numbers of colors $C$. GPIRL was able to perform well even as the number of distractor features increased.

Because of the large number of irrelevant features and the nonlinearity of the reward, this example is particularly challenging for methods that learn linear reward functions. With 16 or more examples, GPIRL consistently learned reward functions that performed as well as the true reward, as shown in Figure 1, and was able to sustain this performance as the number of distractors increased, as shown in Figure 2. While the performance of MaxEnt and FIRL also improved with additional examples, they were consistently outperformed by GPIRL. In the case of FIRL, this was likely due to the suboptimal expert examples. In the case of MaxEnt, although the Laplace prior improved the results, the inability to represent nonlinear rewards limited the algorithm's accuracy. These issues are evident in Figure 3, which shows part of a reward function learned by each method. When using continuous features, the performance of MaxEnt suffered even more from the increased nonlinearity of the reward function, while GPIRL maintained a similar level of accuracy.

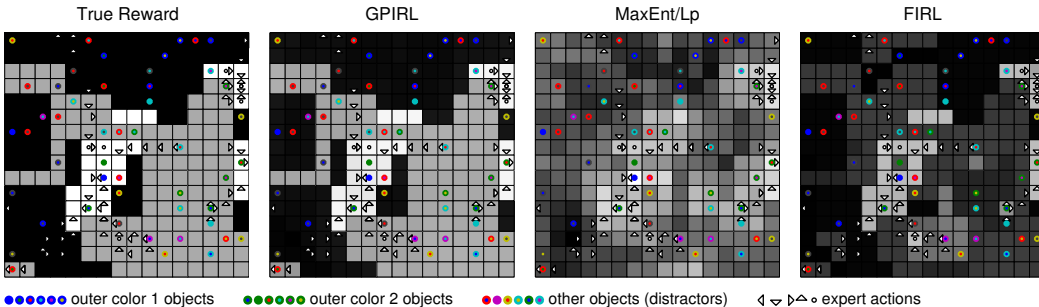

Figure 3: Part of a reward function learned by each algorithm on an objectworld. While GPIRL learned the correct reward function, MaxEnt was unable to represent the nonlinearities, and FIRL learned an overly complex reward under which the suboptimal expert would have been optimal.

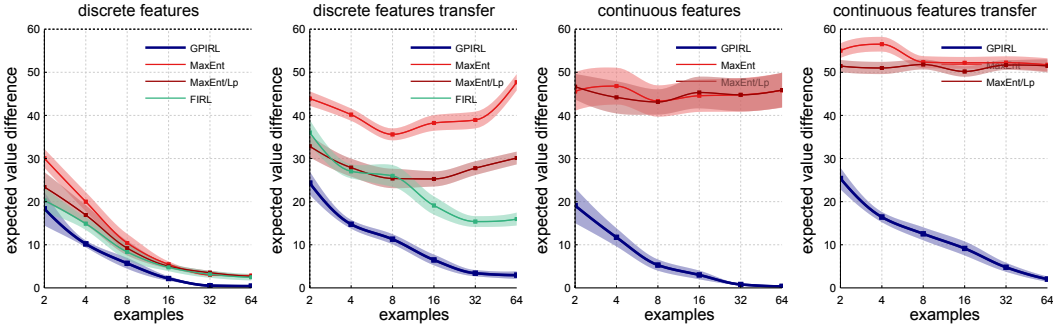

Figure 4: Results for 64-car-length highways with varying example counts. While GPIRL achieved only modest improvement over prior methods on the training environment, the large improvement in the transfer tests indicates that the underlying reward structure was captured more accurately.

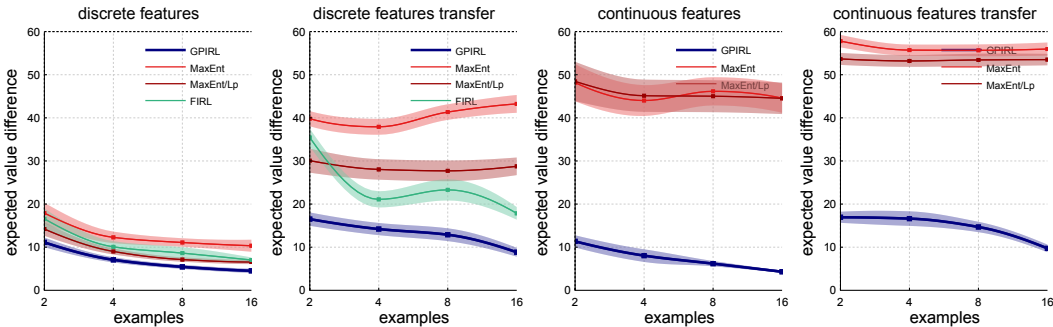

Figure 5: Evaluation on the highway environment with human demonstrations. GPIRL learned a reward function that more accurately reflected the true policy the expert was attempting to emulate.

## 7.2 Highway Driving Behavior

In addition to the objectworld environment, we evaluated the algorithms on more concrete behaviors in the context of a simple highway driving simulator, modeled on the experiment in [5] and similar evaluations in other work [1]. The task is to navigate a car on a three-lane highway, where all other vehicles move at a constant speed. The agent can switch lanes and drive at up to four times the speed of traffic. Other vehicles are either civilian or police, and each vehicle can be a car or motorcycle. Continuous features indicate the distance to the nearest vehicle of a specific class (car or motorcycle) or category (civilian or police) in front of the agent, either in the same lane, the lane to the right, the lane to the left, or any lane. Another set of features gives the distance to the nearest such vehicle in a given lane behind the agent. There are also features to indiciate the current speed and lane. Discrete features again discretize the continuous features, with distances discretized in the same way as in the objectworld. In this section, we present results from synthetic and manmade demonstrations of a policy that drives as fast as possible, but avoids driving more than double the speed of traffic within two car-lengths of a police vehicle. Due to the connection between the police and speed features, the reward for this policy is nonlinear. We also evaluated a second policy that instead avoids driving more than double the speed of traffic in the rightmost lane. The results for this policy were similar to the first, and are included in the supplementary result tables.

Figure 4 shows a comparison of GPIRL and prior algorithms on highways with varying numbers of 32-step synthetic demonstrations of the "police" task. GPIRL only modestly outperformed prior methods on the training environments with discrete features, but achieved large improvement on the transfer experiment. This indicates that, while prior algorithms learned a reasonable reward, this reward was not expressed in terms of the correct features, and did not generalize correctly. With continuous features, the nonlinearity of the reward was further exacerbated, making it difficult for linear methods to represent it even on the training environment. In Figure 5, we also evaluate how GPIRL and prior methods were able to learn the "police" behavior from human demonstrations.

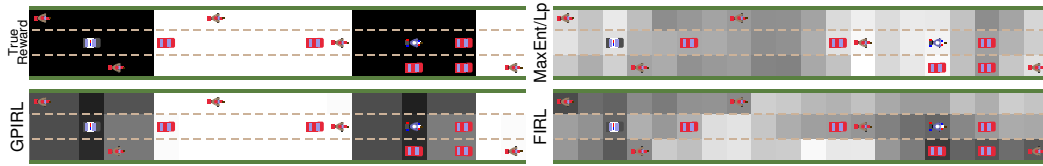

Figure 6: Highway reward functions learned from human demonstration. Road color indicates the reward at the highest speed, when the agent should be penalized for driving fast near police vehicles. The reward learned by GPIRL most closely resembles the true one.

Although the human demonstrations were suboptimal, GPIRL was still able to learn a reward function that reflected the true policy more accurately than prior methods. Furthermore, the similarity of GPIRL's performance with the human and synthetic demonstrations suggests that its model of suboptimal expert behavior is a reasonable reflection of actual human suboptimality. An example of rewards learned from human demonstrations is shown in Figure 6. Example videos of the learned policies and human demonstrations, as well as source code for our implementation of GPIRL, can be found at `http://graphics.stanford.edu/projects/gpirl/index.htm`

# 8   Discussion and Future Work

We presented an algorithm for inverse reinforcement learning that represents nonlinear reward functions with Gaussian processes. Using a probabilistic model of a stochastic expert with a GP prior on reward values, our method is able to recover both a reward function and the hyperparameters of a kernel function that describes the structure of the reward. The learned GP can be used to predict a reward function consistent with the expert on any state space in the domain of the features.

In experiments with nonlinear reward functions, GPIRL consistently outperformed prior methods, especially when generalizing the learned reward to new state spaces. However, like many GP models, the GPIRL log likelihood is multimodal. When using the warped kernel function, a random restart procedure was needed to consistently find a good optimum. More complex kernels might suffer more from local optima, potentially requiring more robust optimization methods.

It should also be noted that our experiments were intentionally chosen to be challenging for algorithms that construct rewards as linear combinations. When good features that form a linear basis for the reward are already known, prior methods such as MaxEnt would be expected to perform comparably to GPIRL. However, it is often difficult to ensure this is the case in practice, and previous work on margin-based methods suggests that nonlinear methods often outperform linear ones [13, 14].

When presented with a novel state space, GPIRL currently uses the mean posterior of the GP to estimate the reward function. In principle, we could leverage the fact that GPs learn distributions over functions to account for the uncertainty about the reward in states that are different from any of the inducing points. For example, such an approach could be used to learn a "conservative" policy that aims to achieve high rewards with some degree of certainty, avoiding regions where the reward distribution has high variance. In an interactive training setting, such a method could also inform the expert about states that have high reward variance and require additional demonstrations.

More generally, by introducing Gaussian processes into inverse reinforcement learning, GPIRL can benefit from the wealth of prior work on Gaussian process regression. For instance, we apply ideas from sparse GP approximation in the use of a small set of inducing points to learn the reward function in time linear in the number of states. A substantial body of prior work discusses techniques for automatically choosing or optimizing these inducing points [8], and such methods could be incorporated into GPIRL to learn reward functions with even smaller active sets. We also demonstrate how different kernels can be used to learn different types of reward structure, and further investigation into the kinds of kernel functions that are useful for IRL is another exciting avenue for future work.

**Acknowledgments.**   We thank Andrew Y. Ng and Krishnamurthy Dvijotham for helpful feedback and discussion. This work was supported by NSF Graduate Research Fellowship DGE-0645962.

## References

[1] P. Abbeel and A. Y. Ng. Apprenticeship learning via inverse reinforcement learning. In *ICML '04: Proceedings of the 21st International Conference on Machine Learning*, 2004.

[2] M. P. Deisenroth, C. E. Rasmussen, and J. Peters. Gaussian process dynamic programming. *Neurocomputing*, 72(7–9):1508–1524, 2009.

[3] K. Dvijotham and E. Todorov. Inverse optimal control with linearly-solvable MDPs. In *ICML '10: Proceedings of the 27th International Conference on Machine Learning*, pages 335–342, 2010.

[4] Y. Engel, S. Mannor, and R. Meir. Reinforcement learning with Gaussian processes. In *ICML '05: Proceedings of the 22nd International Conference on Machine learning*, pages 201–208, 2005.

[5] S. Levine, Z. Popović, and V. Koltun. Feature construction for inverse reinforcement learning. In *Advances in Neural Information Processing Systems 23*. 2010.

[6] G. Neu and C. Szepesvári. Apprenticeship learning using inverse reinforcement learning and gradient methods. In *Uncertainty in Artificial Intelligence (UAI)*, 2007.

[7] A. Y. Ng and S. J. Russell. Algorithms for inverse reinforcement learning. In *ICML '00: Proceedings of the 17th International Conference on Machine Learning*, pages 663–670, 2000.

[8] J. Quiñonero Candela and C. E. Rasmussen. A unifying view of sparse approximate Gaussian process regression. *Journal of Machine Learning Research*, 6:1939–1959, 2005.

[9] D. Ramachandran and E. Amir. Bayesian inverse reinforcement learning. In *IJCAI'07: Proceedings of the 20th International Joint Conference on Artifical Intelligence*, pages 2586–2591, 2007.

[10] C. E. Rasmussen and M. Kuss. Gaussian processes in reinforcement learning. In *Advances in Neural Information Processing Systems 16*, 2003.

[11] C. E. Rasmussen and C. K. I. Williams. *Gaussian Processes for Machine Learning*. The MIT Press, 2005.

[12] N. Ratliff, J. A. Bagnell, and M. A. Zinkevich. Maximum margin planning. In *ICML '06: Proceedings of the 23rd International Conference on Machine Learning*, pages 729–736, 2006.

[13] N. Ratliff, D. Bradley, J. A. Bagnell, and J. Chestnutt. Boosting structured prediction for imitation learning. In *Advances in Neural Information Processing Systems 19*, 2007.

[14] N. Ratliff, D. Silver, and J. A. Bagnell. Learning to search: Functional gradient techniques for imitation learning. *Autonomous Robots*, 27(1):25–53, 2009.

[15] U. Syed and R. Schapire. A game-theoretic approach to apprenticeship learning. In *Advances in Neural Information Processing Systems 20*, 2008.

[16] B. D. Ziebart. *Modeling Purposeful Adaptive Behavior with the Principle of Maximum Causal Entropy*. PhD thesis, Carnegie Mellon University, 2010.

[17] B. D. Ziebart, A. Maas, J. A. Bagnell, and A. K. Dey. Maximum entropy inverse reinforcement learning. In *AAAI Conference on Artificial Intelligence (AAAI 2008)*, pages 1433–1438, 2008.

